# Contour-Map Encoding of Shape for Early Vision

Pentti Kanerva
Research Institute for Advanced Computer Science
Mail Stop 230-5, NASA Ames Research Center
Moffett Field, California  94035

ABSTRACT

Contour maps provide a general method for
recognizing two-dimensional shapes.  All but
blank images give rise to such maps, and people
are good at recognizing objects and shapes
from them.  The maps are encoded easily in
long feature vectors that are suitable for
recognition by an associative memory.  These
properties of contour maps suggest a role for
them in early visual perception.  The prevalence
of direction-sensitive neurons in the visual
cortex of mammals supports this view.

INTRODUCTION

Early vision refers here to the first stages of visual
perception of an experienced (adult human) observer.
Overall, visual perception results in the identification of
what is being viewed:  We recognize an image as the letter A
because it looks to us like other As we have seen.  Early
vision is the beginning of this process of identification--
the making of the first guess.
    Early vision cannot be based on special or salient
features.  For example, we normally think of the letter A
as being composed of two slanted strokes, / and \, meeting
at the top and connected in the middle by a horizontal
stroke, -.  The strokes and their coincidences define all
the features of A.  However, we recognize the As in Figure 1
even though the strokes and the features, if present at all,
do not stand out in the images.

Most telling about human vision is that we can recognize such As after seeing more or less normal As only.  The challenge of early vision, then, is to find general encoding mechanisms that turn these quite dissimilar images of the same object into similar internal representations while leaving the representations of different objects dissimilar; and to find basic pattern-recognition mechanisms that work with these representations.  Since our main work is on associative memories, we have been interested in ways to encode images into long feature vectors suitable for such memories.  The contour-map method of this paper encodes a variety of images into vectors for associative memories.

## REPRESENTING AN IMAGE AS A CONTOUR MAP

Images take many forms:  line drawings, silhouettes, outlines, dot-matrix pictures, gray-scale pictures, color pictures, and the like, and pictures that combine all these elements.  Common to all is that they occupy a region of (two-dimensional) space.  An early representation of an image should therefore be concerned with how the image controls its space or, in technical terms, how might it be represented as a field.

Let us consider first a gray-scale image.  It defines a field by how dark it is in different places (image intensity--a scalar field--the image itself is the field).  A related field is given by how the darkness changes from place to place (gradient of intensity--a vector field).  Neither one is quite right for recognizing As because reversing the field (turning dark to light and light to dark) leaves us with the "same" A.  However, the dark-and-light reversal leaves the contour lines of the image unchanged (i.e., lines of uniform intensity--technically a tangent field perpendicular to the gradient field).  My proposal is to base initial recognition on the contour lines.

In line drawings and black-and-white images, which have only two darkness levels or "colors", the contour lines are not well defined.  This is overcome by propagating the lines and the edges of the image outward and inward over areas of

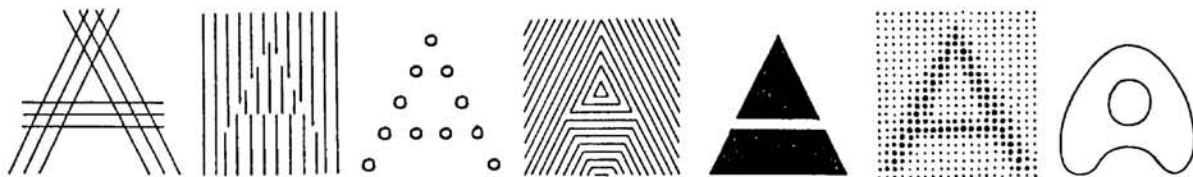

FIGURE 1.  Various kinds of As.

uniform image intensity, in the manner of contour lines, roughly parallel to the lines and the edges. Figure 2 shows only a few such lines, but, in fact, the image is covered with them, running roughly parallel to each other. As a rule, exactly one contour line runs through any given point. Computing its direction is discussed near the end of the paper.

ENCODING THE CONTOUR MAP

Table 1 shows how the direction of the contour at a point can be encoded in three trits (-1, 0, 1 ternary variables). The code divides 180 degrees into six equal sectors and assigns a codeword to each sector. The distance between two codewords is the number of (Hamming) units by which the words differ (L1 distance). The code is circular, and the distance between codewords is related directly to the difference in direction: Directions 30, 60, and 90 degrees apart are encoded with words that are 2, 4, and 6 units apart, respectively. The code wraps around, as do tangents, so that directions 180 degrees apart are encoded the same. For finer discrimination we would use some finer circular code. The zero-word 000, which is equally far from all other words in the code, is used for points at which the direction of the contour is ill-defined, such as the very centers of circles.

This encoding makes the direction of the contour at any point on a map into a three-component vector. To encode the entire map, the vector field is sampled at a fixed, finite set of points, and the encodings of the sample points are concatenated in fixed order into a long vector. In preliminary studies we have used small sample sizes: 7 x 5 (= 35) sample points, each encoded into three trits, for a total vector of (3 x 35 =) 105 trits, and 8 x 8 sample points by three trits for a total vector of 192 trits.

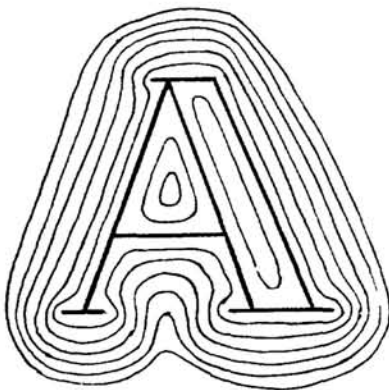

FIGURE 2. Propagating the contour.

For an example, Figure 3 shows the digit 4 drawn on a 21-by-15-pixel grid. It also shows a 7 x 5 sampling grid laid over the image and the direction of the contour at the sample points (shown by short line segments). Below the image are the three-trit encodings of the sample points starting at the upper left corner and progressing by rows, concatenated into a 105-trit encoding of the entire image. In this encoding, + means +1 and - means -1.

### From Positions of the Code to Directional Sensors

Each position of the three-trit code can be thought of as a directional sensor. For example, the center position senses contours at 90 degrees, plus or minus 45 degrees: It is 1 when the direction of the contour is closer to vertical than to horizontal (see Table 1). Similarly, each position of the long (105-trit) code for the entire map can be thought of as a sensor for a specific direction--plus or minus--at a specific location on the map.

An array of sensors will thus encode an image. The sensors are like the direction-sensitive cells of the visual cortex. Such cells, of course, are not laid down with perfect regularity over the cortex, but that does not mean

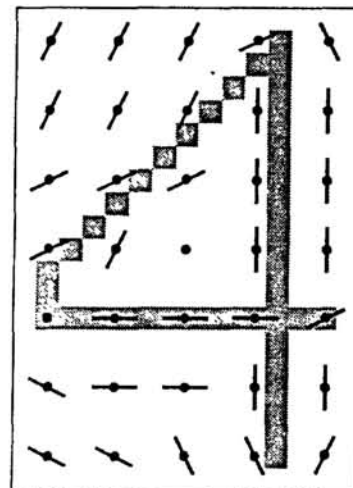

TABLE 1

Coarse Circular Code for
  Direction of Contour
========================

| Direction, degrees | Codeword | | |
|---|---|---|---|
| 0 ± 15 | 1 | -1 | 1 |
| 30 ± 15 | -1 | -1 | 1 |
| 60 ± 15 | -1 | 1 | 1 |
| | | | |
| 90 ± 15 | -1 | 1 | -1 |
| 120 ± 15 | 1 | 1 | -1 |
| 150 ± 15 | 1 | -1 | -1 |
| | | | |
| 180 ± 15 | 1 | -1 | 1 |
| . . . | . . . | | |
| Undefined | 0 | 0 | 0 |

========================

```
-++ -++ -++ --+ ++-
-++ -++ -++ -+- -+-
--+ --+ --+ -+- -+-
--+ -++ 000 -+- -+-
000 +-+ +-+ +-+ --+
+-- +-+ +-+ -+- -+-
+-- +-- ++- ++- -++
```

FIGURE 3.   Encoding an image.

that they could not perform as encoders. Accordingly, a
direction-sensitive cell can be thought of as a feature
detector that encodes for a certain direction at a certain
location in the visual or attentional field. An irregular
array of randomly oriented sensors laid over images would
produce perfectly good encodings of their contour maps.

COMPARING TWO CONTOUR MAPS

How closely do two contour maps resemble each other? For
simplicity, we will compare maps of equal size (and shape)
only. The maps are compared point to point. The difference
at a point is the difference in the direction of the contour
at that point on the two maps--that is, the magnitude of the
lesser of the two angles made by the two contour lines that
run through the two points that correspond to each other
on the two maps. The maximum difference at a point is
therefore 90 degrees. The entire maps are then compared
by adding the pointwise differences over all the points (by
integrating over the area of the map).

The purpose of the encoding is to make the comparing of
maps simple. The code is so constructed that the difference
of two maps at a point is roughly proportional to the
distance between the two (3-trit) codewords--one from each
map--for that point. We need not even concern ourselves
with the finding of the lesser of the two angles made by the
crossing of the two contours; the distance between codewords
accounts for that automatically.

Entire maps are then compared by adding together the
distances at the (35) sample points. This is equivalent
to computing the distance between the (105-trit) codewords
for the two maps. This distance is proportional to the
difference between the maps, and it is approximately so
because the maps are sampled at a small number of points
and because the direction at each point is coded coarsely.

COMPUTING THE DIRECTION OF THE CONTOUR

We have not explored widely how to compute contours from
images and merely outline here one method, not exactly
biological, that works for line drawings and two-tone images
and that can be generalized to gray-scale images and even
to many multicolor images. We have also experimented with
oriented, difference-of-Gaussian filters of Parent and
Zucker (1985) and with cortex transforms of Watson (1987).

The contours are based on a simple model of attraction,
akin to gravity, by assuming that the lines and the edges
of the image attract according to their distance from the
point. The net attraction at any point on the image defines

a gradient field, and the contours are perpendicular to it.

In practice we work with pixels and assume, for the sake of the gravity model, that pixels of the same color--same as that of the sample point P for which we are computing the direction--have mass zero and those of the opposite color have mass one. For the direction to be independent of scale, the attractive force must be inversely proportional to some power of the distance. Powers greater than 2 make the computation local. For example, power 7 means that one pixel, twice as far as another, contributes only 1/128 as much as the other to the net force. To make the attraction somewhat insensitive to noise, a small constant, 3, is added to the distance. (The values 7 and 3 were chosed after a small amount of experimentation.) Hence, pixel X (of mass 1) attracts P with a magnitude

$$[d(P,X) + 3]^{-7}$$

force in the direction of X, where d(P,X) is the (Euclidean) distance between P and X. The vector sum of the forces over all pixels X (of mass 1) then is the attractive force at point P, and the direction of the contour at P is perpendicular to it. The magnitude of the vector sum is scaled by dividing it with the sum of the magnitudes of its components. This scaled magnitude indicates how well the direction is defined in the image.

When this computation is made at a point on a (one-pixel wide) line, the result is a zero-vector (the gradient at the top of a ridge is zero). However, we want to use the direction of the line itself as the direction of the contour. To this end, we compute at each sample point P another vector that detects linear features, such as lines. This computation is based on the above attraction model, modified as follows: Pixels of the same color as P's now have mass one and those of the opposite color have mass zero (the pixel at P being always regarded as having mass zero); and the direction of the force, instead of being the angle from P to X, is twice that angle. The doubling of the angle makes attractive forces in opposite directions (along a line) reenforce each other and in perpendicular directions cancel out each other. The angle of the net force is then halved, and the magnitude of the force is scaled as above.

The two computations yield two vectors, both representing the direction of the contour at a point. They can be combined into a single vector by doubling their angles, to eliminate 180-degree ambiguities, by adding together the resulting vectors, and by halving the angle of the sum. The direction of the result gives the direction of the contour, and the magnitude of the result indicates how well

this direction is defined.  If the magnitude is below some
threshold, the direction is taken to be undefined and is
encoded with 000.

SOME COMPARISONS

The method is very general, which is at once its virtue and
limitation.  The virtue is that it works where more specific
methods fail, the limitation that the specific methods are
needed for specific problems.

In our preliminary experiments with handwritten Zip-code
digits, low-pass filtering (blurring) an image, as a method
of encoding it, and contour maps resulted in similar rates
of recognition by a sparse distributed memory.  Higher rates
on this same task were gotten by Denker et al. (1989) by
encoding the image in terms of features specific to
handwriting.

To get an idea of the generality of contour maps, Figure
4 shows encoded maps of ten normal digits like that in
Figure 3, and for three unusual digits barely recognizable
by humans.  The labels for the unusual ones and for their
maps, 8a, 8b, and 9a, tell what digits they were intented
to be.  Table 2 of distances between the encoded maps
shows that 8 gives only the second best match to 8a and 8b,
whereas the digit closest to 9a indeed is 9.  This suggest
that a system trained on normal letters and digits would do

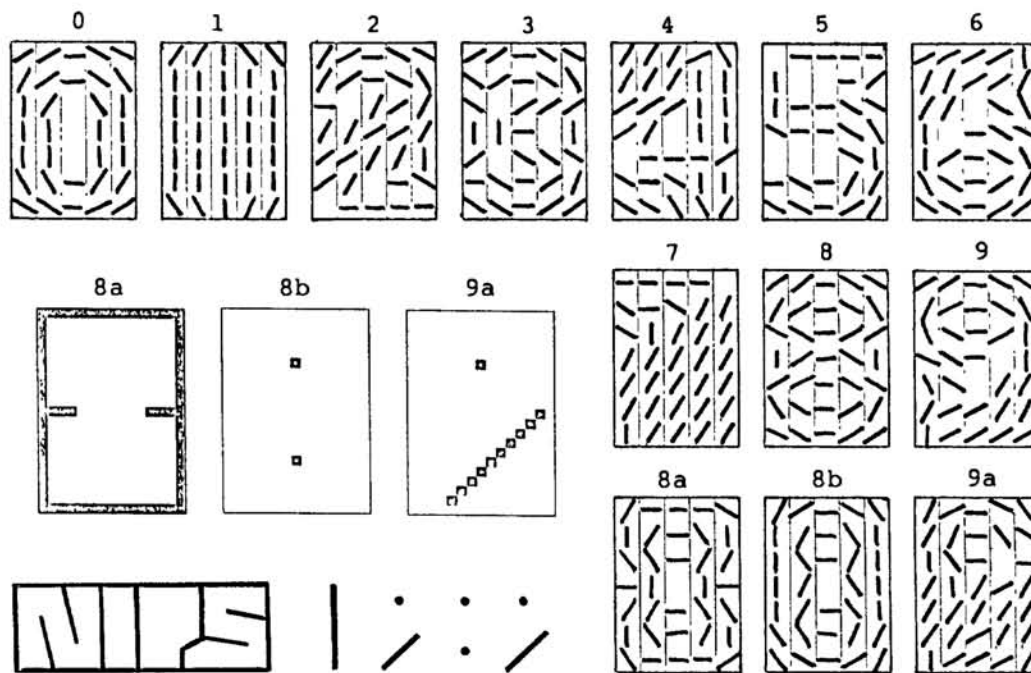

FIGURE 4.  Contour maps of digits.  Unusual text.

TABLE 2
Distances Between Normal and Unusual Digits of Figure 4

| | 0 | 1 | 2 | 3 | 4 | 5 | 6 | 7 | 8 | 9 |
|---|---|---|---|---|---|---|---|---|---|---|
| 8a | 62 | 95 | 80 | 74 | 91 | 87 | 83 | 86 | 67 | 79 |
| 8b | 38 | 71 | 88 | 64 | 77 | 73 | 65 | 88 | 51 | 73 |
| 9a | 70 | 89 | 66 | 90 | 109 | 99 | 103 | 62 | 83 | 59 |

a fair job at recognizing the 'NIPS 1989' at the bottom of Figure 4. Systems that encode characters as bit maps, or that take them as composed of strokes, likewise trained, would not do nearly as well. Going back to the As of Figure 1, they can, with one exception, be recognized based on the map of a normal A. Logograms are a rich source of images of this kind. They are excellent for testing a vision system for generality. Finally, other oriented fields, not just contour maps, can be encoded with methods similar to this for recognition by an associative memory.

## Acknowledgements

This research was supported by the National Aeronautics and Space Administration (NASA) with cooperative agreement No. NCC2-387 with the Universities Space Research Association. The idea of contour maps was inspired by the gridfonts of Douglas Hofstadter (1985). The first experiments with the contour-map method were done by Bruno Olshausen. The gravity model arose from discussions with Lauri Kanerva. David Rogers made the computer-drawn illustrations.

## References

Denker, J.S., Gardner, W.R., Graf, H.P., Henderson, D., Howard, R.E., Hubbard, W., Jackel, L.D., Baird, H.S., and Guyon, I. (1989) Neural Network Recognizer for Hand-Written Zip Code Digits. In D.S. Touretzky (ed.), Advances in Neural Information Systems, Volume I. San Mateo, California: Kaufmann. 323-331.

Hofstadter, D.R. (1985) Metamagical Themas. New Your: Basic Books.

Parent, P., and Zucker, S.W. (1985) Trace Inference, Curvature Consistency, and Curve Detection. Report CIM-86-3, McGill Research Center for Intelligent Machines, Montreal, Canada.

Watson, A.W. (1987) The Cortex Transform: Rapid Computation of Simulated Neural Images. Computer Vision, Graphics, and Image Processing 39(3):311-327.